# Learning Non-Rigid 3D Shape from 2D Motion

**Lorenzo Torresani**
Stanford University
`ltorresa@cs.stanford.edu`

**Aaron Hertzmann**
University of Toronto
`hertzman@dgp.toronto.edu`

**Christoph Bregler**
New York University
`chris.bregler@nyu.edu`

## Abstract

This paper presents an algorithm for learning the time-varying shape of a non-rigid 3D object from uncalibrated 2D tracking data. We model shape motion as a rigid component (rotation and translation) combined with a non-rigid deformation. Reconstruction is ill-posed if arbitrary deformations are allowed. We constrain the problem by assuming that the object shape at each time instant is drawn from a Gaussian distribution. Based on this assumption, the algorithm simultaneously estimates 3D shape and motion for each time frame, learns the parameters of the Gaussian, and robustly fills-in missing data points. We then extend the algorithm to model temporal smoothness in object shape, thus allowing it to handle severe cases of missing data.

## 1  Introduction

We can generally think of a non-rigid object's motion as consisting of a rigid component plus a non-rigid deformation. For example, a person's head can move rigidly (e.g. turning left or right) while deforming (due to changing facial expressions). If we view this non-rigid motion from a single camera view, the shape and motion are ambiguous: for any hypothetical rigid motion, a corresponding 3D shape can be devised that fits the image observations. Even if camera calibration and rigid motion are known, a depth ambiguity remains. Despite this apparent ambiguity, humans interpret the shape and motion of non-rigid objects with relative ease; clearly, more assumptions about the nature of the deformations are used by humans.

This paper addresses the question: how can we resolve the ambiguity, with as weak assumptions as possible? We argue that, by assuming that the 3D shape is drawn from some non-uniform PDF, we can reconstruct 3D non-rigid shape from 2D motion unambiguously. Moreover, we show that this can be done *without* assuming that the parameters of the PDF are known in advance. The use of a proper PDF makes the technique robust to noise and overfitting. We demonstrate this approach by modeling the PDF as a Gaussian distribution (more specifically, as a factor analyzer), and describe a novel EM algorithm for simultaneously learning the 3D shapes, the rigid motion, and the parameters of the Gaussian. We also generalize this approach by modeling the shape as a Linear Dynamical System (LDS).

Our algorithm can be thought of as a structure-from-motion (SFM) algorithm with a learning component: we assume that a set of labeled point tracks have been extracted from a raw video sequence, and the goal is to estimate 3D shape, camera motion, and a deformation PDF. Our algorithm is well-suited to reconstruction in the case of missing data, such as due to occlusions and other tracking outliers. However, we show significant improvements over previous algorithms even when all tracks are visible.

Our work may also be seen as unifying Active Shape Models [1, 2, 5] with SFM, where both are estimated jointly from an image sequence. Our methods are closely related to factor analysis, probabilistic PCA, and linear dynamical systems. Our missing-data technique can be viewed as generalizing previous algorithms for SFM with missing data (e.g. [8, 9]) to the nonrigid case. In work concurrent to our own, Gruber and Weiss [7] also apply EM to SFM; their work focuses on the rigid case with known noise, and applies temporal smoothing to rigid motion parameters rather than shape.

## 2 Deformation, Shape, and Ambiguities

We now formalize the problem of interpreting non-rigid shape and motion. We assume that a scene consists of $J$ scene points $\mathbf{s}_{j,t}$, where $j$ is an index over scene points, and $t$ is an index over image frames. The 2D projections $\mathbf{p}_{j,t}$ of these points are imaged under orthographic projection:

$$\mathbf{p}_{j,t} = \mathbf{R}_t(\mathbf{s}_{j,t} + \mathbf{d}_t) + n \qquad (1)$$

where $\mathbf{p}_{j,t}$ is the 2D projection of scene point $j$ at time $t$, $\mathbf{d}_t$ is a $2 \times 1$ translation vector, $\mathbf{R}_t$ is a $2 \times 3$ matrix that combines rotation with orthographic projection [12], and $n$ is zero-mean Gaussian noise with variance $\sigma^2$. Collecting the projected points into a $2 \times J$ matrix $\mathbf{P}_t = [\mathbf{p}_{1,t}, ..., \mathbf{p}_{J,t}]$ and the 3D shape into a $3 \times J$ matrix $\mathbf{S}_t = [\mathbf{s}_{1,t}, ...\mathbf{s}_{J,t}]$ gives the equivalent form

$$\mathbf{P}_t = \mathbf{R}_t(\mathbf{S}_t + \mathbf{D}_t) + \mathbf{N} \qquad (2)$$

where $\mathbf{D}_t = \mathbf{d}_t \mathbf{1}^T$ contains $J$ copies of the translation matrix $\mathbf{d}_t$. Note that rigid motion of the object and rigid motion of the camera are interchangeable. Our goal is to estimate the time-varying shape $\mathbf{S}_t$ and motion $(\mathbf{R}_t, \mathbf{D}_t)$ from the observed projections $\mathbf{P}_t$. Without any constraints on the 3D shape $\mathbf{s}_{j,t}$, this problem is extremely ambiguous [11]. For example, given a shape $\mathbf{S}_t$ and motion $(\mathbf{R}_t, \mathbf{D}_t)$ and an arbitrary orthonormal matrix $\mathbf{A}_t$, we can produce a new shape $\mathbf{A}_t\mathbf{S}_t$ and motion $(\mathbf{R}_t\mathbf{A}_t^{-1}, \mathbf{A}_t\mathbf{D}_t)$ that together give identical 2D projections as the original model, even if a different matrix $\mathbf{A}_t$ is applied in every frame.

A common way to model non-rigid deformations is to assume that the shape is produced by adding deformations to a *shape average* $\bar{\mathbf{S}}$:

$$\mathbf{S}_t = \bar{\mathbf{S}} + \sum_{k=1}^{K} \mathbf{V}_k z_{k,t} \qquad (3)$$

where $z_{k,t}$ are scalar per-frame weights that indicate the contributions of the deformations to each shape; these weights are combined in a vector $\mathbf{z}_t = [z_{1,t}, ..., z_{K,t}]^T$. Together, $\bar{\mathbf{S}}$ and $\mathbf{V}_k$ are referred to as the *shape basis*. Equivalently, the space of possible shapes may be described by linear combinations of *basis shapes*, by selecting $K + 1$ linearly independent points in the space. This model was first applied to non-rigid SFM by Bregler et al. [4]. However, this model contains ambiguities, since, for some 3D shape and motion, there will still be ways to combine different weights and a different rigid motion to produce the same 3D shape. Since we are performing a 2D projection, an additional depth ambiguity occurs. For example, whenever there exist weights $w_k$ such that $\mathbf{R}_t \sum \mathbf{V}_k w_k = 0$ and $\sum \mathbf{V}_k w_k \neq 0$, these weights define a linear space of distinct 3D shapes (with weights

$z_{t,k} + \alpha w_k$) that give identical 2D projections. (When the number of basis shapes is small, these ambiguities are rarer and may not make a dramatic impact.) Furthermore, a least-squares fit may overfit noise, especially with many basis shapes. As the number of basis shapes grows, the problem is more likely to become unconstrained, eventually approaching the totally unconstrained case described above.

The ambiguity and overfitting may be resolved by introducing regularization terms that penalize large deformations, and then solving for 3D shape in a least-squares sense. Soatto and Yezzi [11] use a regularization term equivalent to $\sum_t ||\mathbf{S}_t - \bar{\mathbf{S}}||^2$. However, this regularization may be too restrictive in many cases and too loose in others. For example, when tracking a face, deformations of the jaw are much more likely than deformations of the nose. Moreover, the weight for this regularization term must be specified by hand[1]. Alternatively, Brand [3] proposes placing a user-specified Gaussian prior on the deformation basis and a prior on the deformations based on an initial estimate.

In order to motivate our approach, we can restate the above techniques as follows. Suppose we assume that shapes $\mathbf{S}_t$ are drawn from a probability distribution $p(\mathbf{S}_t|\theta)$ with known parameters $\theta$. The non-rigid shape and motion are estimated by maximizing

$$
\begin{aligned}
p(\mathbf{S}, \mathbf{R}, \mathbf{D}|\mathbf{P}, \theta, \sigma^2) &\propto p(\mathbf{P}|\mathbf{S}, \mathbf{R}, \mathbf{D}, \theta, \sigma^2)p(\mathbf{S}, \mathbf{R}, \mathbf{D}|\theta, \sigma^2) &(4)\\
&\propto \prod_t p(\mathbf{P}_t|\mathbf{S}_t, \mathbf{R}_t, \mathbf{D}_t, \sigma^2)p(\mathbf{S}_t|\theta) &(5)
\end{aligned}
$$

assuming uniform priors on $\mathbf{R}_t$, and $\mathbf{D}_t$. The projection likelihood $p(\mathbf{P}_t|\mathbf{S}_t, \mathbf{R}_t, \mathbf{D}_t, \sigma^2)$ is a spherical Gaussian (Equation 2). The negative log-posterior $-\ln p(\mathbf{S}, \mathbf{R}, \mathbf{D}, \theta|\mathbf{P})$ corresponds to a standard least-squares formulation for SFM, plus a regularization term $-\ln p(\mathbf{S}_t|\theta)$. If we set $p(\mathbf{S}_t|\theta)$ to be a uniform distribution, then we get the highly underconstrained case described above. If we set $p(\mathbf{S}_t|\theta)$ to be a spherical Gaussian with a specified variance (e.g. $p(\mathbf{S}_t|\theta) = \mathcal{N}(\bar{\mathbf{S}}; \sigma^2\mathbf{I})$) then we obtain the simple regularization used previously — the problem is constrained, but by a weak regularization term with a user-specified weight (variance).

**Our approach.** Our approach is to simultaneously estimate the rigid motion and learn the shape PDF. In other words, we estimate $\mathbf{R}, \mathbf{D}, \theta$, and $\sigma^2$ to maximize

$$
\begin{aligned}
p(\mathbf{R}, \mathbf{D}, \theta, \sigma^2|\mathbf{P}) &= \int p(\mathbf{R}, \mathbf{D}, \theta, \mathbf{S}, \sigma^2|\mathbf{P})d\mathbf{S} &(6)\\
&\propto \int p(\mathbf{P}|\mathbf{R}, \mathbf{D}, \mathbf{S}, \sigma^2)p(\mathbf{S}|\theta)d\mathbf{S} &(7)
\end{aligned}
$$

The key idea is that we can estimate shape and motion while learning the parameters of the PDF $p(\mathbf{S}|\theta)$ over shapes. (Our method marginalizes over the unknown shapes $\mathbf{S}_t$, rather than solving for estimates of shape.) In effect, the regularization terms (i.e. the PDF) are learned simultaneously with the rest of SFM. This means that the regularization terms need not be set manually, and can thus be much more sophisticated and have many more parameters than previous methods. In practice, we find that this leads to significantly improved reconstructions over user-specified shape PDFs. We demonstrate the approach by modeling the shape PDF as a general Gaussian. We reduce the dimensionality of the Gaussian by representing it as a factor analyzer. In this case, the factors $\mathbf{V}_k$ may be interpreted as basis deformations. We later generalize this approach to model shape as an LDS, leading to temporal correlations in the shape PDF.

It might seem that, since the parameters of the PDF are not known *a priori*, the algorithm could estimate wildly varying shapes, and then learn a correspondingly spread-out PDF.

However, such a spread-out PDF would assign very low likelihood to the solution and thus be suboptimal; this is a typical case of Bayesian learning naturally balancing the desire to fit the data with the desire for a "simple" model. One way to see this is to consider the terms of $-\ln p(\mathbf{R}, \mathbf{D}, \theta | \mathbf{P})$ in the case of the Gaussian prior PDF: in addition to the data-fitting term and the regularization term, there is a "normalization constant" term of $T \ln |\phi|$, where $T$ is the number of frames and $\phi$ is the covariance of the shape PDF. This term directly penalizes spread-out Gaussians. Hence, the optimal solution trades-off between (a) fitting the projection data, (b) fitting the shapes $\mathbf{S}_t$ to the shape PDF (regularizing), and (c) minimizing the variance of the shape PDF as much as possible. The algorithm simultaneously regularizes and learns the regularization.

## 3  Learning a Gaussian shape distribution

We now describe our algorithm in detail. We model $p(\mathbf{S}_t|\theta)$ as a factor analyzer [6]. In this setting, the factors of the Gaussian can be interpreted as basis deformations — shape is modeled by Equation 3 — but the weights $\mathbf{z}_t$ are now hidden variables, with zero-mean Gaussian priors with unit variance for each:

$$\mathbf{z}_t \quad \sim \quad \mathcal{N}(0; \mathbf{I}) \tag{8}$$

The shape and projection model is then completely specified by Equations 2, 3, and 8. The problem of non-rigid SFM is now to solve for the maximum likelihood estimates of $\mathbf{R}_t, \mathbf{D}_t, \bar{\mathbf{S}}, \mathbf{V},$ and $\sigma^2$, i.e. maximize $p(\mathbf{R}_t, \mathbf{D}_t, \bar{\mathbf{S}}, \mathbf{V}, \sigma^2 | \mathbf{P}_t) \propto \prod_t p(\mathbf{P}_t | \mathbf{R}_t, \mathbf{D}_t, \bar{\mathbf{S}}, \mathbf{V}, \sigma^2) = \prod_t \int p(\mathbf{P}_t, \mathbf{z}_t | \mathbf{R}_t, \mathbf{D}_t, \bar{\mathbf{S}}, \mathbf{V}, \sigma^2) p(\mathbf{z}_t) d\mathbf{z}_t$

### 3.1  Vectorized form.

For later computations, it is useful to rewrite the model in a vectorized form. First, define $\mathbf{f}_t$ to be the vector of point tracks $\mathbf{f}_t = \mathrm{vec}(\mathbf{P}_t) = [x_{1,t}, y_{1,t}, ..., x_{J,t}, y_{J,t}]^T$. Note that $\mathbf{f}_t$ is the same variable as $\mathbf{P}_t$, but written as a vector rather than a matrix[2]. Expanding $\mathbf{f}_t$ we have

$$\mathbf{f}_t \quad = \quad \mathrm{vec}(\mathbf{P}_t) = \mathrm{vec}(\mathbf{R}_t \mathbf{S}_t + \mathbf{R}_t \mathbf{D}_t + \mathbf{N}_t) \tag{9}$$

$$= \quad \sum_{k=1}^K \mathrm{vec}(\mathbf{R}_t \mathbf{V}_k) z_{k,t} + \mathrm{vec}(\mathbf{R}_t \bar{\mathbf{S}}) + \mathrm{vec}(\mathbf{R}_t \mathbf{D}_t) + \mathrm{vec}(\mathbf{N}_t) \tag{10}$$

$$= \quad \mathbf{M}_t \mathbf{z}_t + \bar{\mathbf{f}}_t + \mathbf{T}_t + \mathrm{vec}(\mathbf{N}_t) \tag{11}$$

where $\mathbf{M}_t = [\mathrm{vec}(\mathbf{R}_t \mathbf{V}_1), ..., \mathrm{vec}(\mathbf{R}_t \mathbf{V}_K)]$, $\mathbf{z}_t = [z_{1,t}, ..., z_{K,t}]^T$, $\bar{\mathbf{f}}_t = \mathrm{vec}(\mathbf{R}_t \bar{\mathbf{S}})$ and $\mathbf{T}_t = \mathrm{vec}(\mathbf{R}_t \mathbf{D}_t) = [(\mathbf{R}_t \mathbf{d}_t)^T, ..., (\mathbf{R}_t \mathbf{d}_t)^T]^T = [\mathbf{t}_t^T, ..., \mathbf{t}_t^T]^T$. Note that the marginal distribution over shape — as well as its projection — is Gaussian:

$$p(\mathbf{f}_t | \psi) \quad = \quad \int p(\mathbf{f}_t | \mathbf{z}_t, \psi) p(\mathbf{z}_t | \psi) d\mathbf{z}_t \tag{12}$$

$$= \quad \mathcal{N}(\mathbf{f}_t | \mathbf{T}_t + \bar{\mathbf{f}}_t; \mathbf{M}_t \mathbf{M}_t^T + \sigma^2 \mathbf{I}) \tag{13}$$

where $\psi$ encapsulates the model parameters $\bar{\mathbf{S}}, \mathbf{V}_k, \mathbf{R}_t, \mathbf{D}_t$ and $\sigma^2$.

Let $\tilde{\mathbf{H}} = [\mathrm{vec}(\bar{\mathbf{S}}), \mathrm{vec}(\mathbf{V}_1), ..., \mathrm{vec}(\mathbf{V}_K)]$ and $\tilde{\mathbf{z}}_t = [1, \mathbf{z}_t^T]^T$. We can also rewrite the shape equation as $\mathrm{vec}(\mathbf{R}_t \mathbf{S}_t) = (\mathbf{I} \otimes \mathbf{R}_t) \mathrm{vec}(\mathbf{S}_t) = (\mathbf{I} \otimes \mathbf{R}_t) \tilde{\mathbf{H}} \tilde{\mathbf{z}}_t$, by using the identity $\mathrm{vec}(\mathbf{ABC}) = (\mathbf{C}^T \otimes \mathbf{A}) \mathrm{vec}(\mathbf{B})$. The symbol $\otimes$ denotes Kronecker product.

### 3.2  Generalized EM algorithm.

Given a set of point tracks $\mathbf{P}$ (equivalently, $\mathbf{f}$), we can estimate the motion and deformation model using EM; the algorithm is similar to EM for factor analysis [6].

**The E-step.**  We estimate the distribution over $\mathbf{z}_t$ given the current motion and shape estimates, for each frame $t$. Defining $q(\mathbf{z}_t)$ to be the distribution to be estimated in frame $t$, it can be computed as

$$
\begin{aligned}
q(\mathbf{z}_t) &= p(\mathbf{z}_t | \mathbf{f}_t, \psi) &\quad (14)\\
&= \mathcal{N}(\mathbf{z}_t | \beta(\mathbf{f}_t - \bar{\mathbf{f}}_t - \mathbf{T}_t); \mathbf{I} - \beta \mathbf{M}_t) &\quad (15)\\
\beta &= \mathbf{M}_t^T (\mathbf{M}_t \mathbf{M}_t^T + \sigma^2 \mathbf{I})^{-1} &\quad (16)
\end{aligned}
$$

The matrix inversion lemma may be used to accelerate the computation of $\beta$. We define the expectations $\mu_t \equiv E_q[\mathbf{z}_t]$ and $\phi_t \equiv E_q[\mathbf{z}_t \mathbf{z}_t^T]$ and compute them as:

$$
\begin{aligned}
\mu_t &= \beta(\mathbf{f}_t - \bar{\mathbf{f}}_t - \mathbf{T}_t) &\quad (17)\\
\phi_t &= \mathbf{I} - \beta \mathbf{M}_t + \mu_t \mu_t^T &\quad (18)
\end{aligned}
$$

We also define $\tilde{\mu}_t = E[\tilde{\mathbf{z}}_t] = [1, \mu_t^T]^T$ and $\tilde{\phi} = E[\tilde{\mathbf{z}}_t \tilde{\mathbf{z}}_t^T] = \begin{bmatrix} 1 & \mu_t^T \\ \mu_t & \phi_t \end{bmatrix}$.

**The M-step.**  We estimate the motion parameters by minimizing

$$
\begin{aligned}
Q(\mathbf{P}, \psi) &= E_{q(\mathbf{z}_1), \dots, q(\mathbf{z}_T)}[-\log p(\mathbf{P}|\psi)] &\quad (19)\\
&= \sum_t E_{q(\mathbf{z}_t)}[\|\mathbf{f}_t - \mathrm{vec}(\mathbf{R}_t \mathbf{S}_t) - \mathbf{T}_t\|^2 / (2\sigma^2)] + 2JT \log \sqrt{2\pi\sigma^2} &\quad (20)
\end{aligned}
$$

This function is quadratic in the shape parameters $(\bar{\mathbf{S}}, \mathbf{V}_k)$, in the rigid motion parameters $(\mathbf{R}_t, \mathbf{T}_t)$ and in the gaussian noise variance parameter $\sigma^2$. To update each of these parameters we compute the corresponding partial derivative of the expected log likelihood, set it to zero and solve it. The parameter update rules are:

- Shape basis:

$$
\mathrm{vec}(\tilde{\mathbf{H}}) \leftarrow \left( \sum_t (\tilde{\phi}_t \otimes (\mathbf{I} \otimes \mathbf{R}_t^T \mathbf{R}_t)) \right)^{-1} \mathrm{vec}\left( \sum_t (\mathbf{I} \otimes \mathbf{R}_t)^T (\mathbf{f}_t - \mathbf{T}_t)\tilde{\mu}_t^T \right) \tag{21}
$$

- Noise variance:

$$
\sigma^2 \leftarrow \frac{1}{2JT} \sum_t (\|\mathbf{f}_t - \bar{\mathbf{f}}_t - \mathbf{T}_t\|^2 - 2(\mathbf{f}_t - \bar{\mathbf{f}}_t - \mathbf{T}_t)^T \mathbf{M}_t \mu_t + \mathrm{tr}(\mathbf{M}_t^T \mathbf{M}_t \phi_t)) \tag{22}
$$

- Translation:

$$
\mathbf{T}_t \leftarrow (\mathbf{1} \otimes \mathbf{I}) \frac{1}{J} \sum_j (\mathbf{f}_{tj} - \mathbf{R}_t(\bar{\mathbf{S}}_j + \sum_k \mathbf{V}_{kj} \mu_{tk})) \tag{23}
$$

- Rotation:

$$
\mathbf{R}_t \leftarrow \arg\min_{R_t} \| \mathbf{R}_t \sum_j (\tilde{\mathbf{H}}_j \tilde{\phi}_t \tilde{\mathbf{H}}_j^T) - \sum_j ((\mathbf{f}_{tj} - \mathbf{t}_t)\tilde{\mu}_t^T \tilde{\mathbf{H}}_j^T)\| \tag{24}
$$

where $\tilde{\mathbf{H}} = [\tilde{\mathbf{H}}_1^T, \dots, \tilde{\mathbf{H}}_J^T]^T$ and $\mathbf{f}_t = [\mathbf{f}_{t1}, \dots, \mathbf{f}_{tJ}]$.

Since the system of equations in Equation 21 is large and sparse, we solve it using conjugate gradient. In Equation 24, we enforce orthonormality of rotations by parameterizing $\mathbf{R}_t$ with exponential coordinates. We linearize the equation with respect to the exponential coordinates, and solve the resulting quadratic.

If any of the point tracks are missing, they are also filled in during the M-step. Let $\mathbf{f}_t^*$ denote the elements of a frame of tracking data that are not observed; they are estimated as

$$\mathbf{f}_t^* \quad \leftarrow \quad \bar{\mathbf{f}}_t^* + \mathbf{M}_t^* \mu_t + \mathbf{T}_t^* \tag{25}$$

where $(^*)$ indicates rows that correspond to the missing data.

In our M-step, we apply each of these updates once, although they could also be alternated. Once EM has converged, the maximum likelihood shapes may be computed as $\mathbf{S}_t = \bar{\mathbf{S}} + \sum_k \mathbf{V}_k \mu_{t,k}$.

## 4   Learning dynamics

Many real deformations contain some temporal smoothness. We model temporal behavior of deformations using a Linear Dynamical System (LDS). In this model, Equation 8 is replaced with

$$\mathbf{z}_0 \quad \sim \quad \mathcal{N}(0; \mathbf{I}) \tag{26}$$
$$\mathbf{z}_t \quad = \quad \mathbf{\Phi}\mathbf{z}_{t-1} + \mathbf{n}, \quad \mathbf{n} \sim \mathcal{N}(0; \mathbf{Q}) \tag{27}$$

where $\mathbf{\Phi}$ is an arbitrary unknown $K \times K$ matrix, and $\mathbf{Q}$ is a $K \times K$ covariance matrix. For certain estimates of $\mathbf{\Phi}$, this model corresponds to an assumption of continuously or slowly changing shape. Since our model is a special form of Shumway and Stoffer's algorithm for LDS learning with EM [10], it is straightforward to adapt it to our needs. In the E-step, we apply Shumway and Stoffer's E-step to estimate $\mu_t, \phi_t$, and $E[\mathbf{z}_t \mathbf{z}_{t-1}^T]$, based on $\mathbf{P}_t, \bar{\mathbf{S}}, \mathbf{M}_t, \mathbf{\Phi}, \mathbf{Q}$, and $\sigma^2$. In the M-step, we apply the same shape and motion updates as in the previous section; additionally, we update $\mathbf{\Phi}$ and $\mathbf{Q}$ in the same way as in Shumway and Stoffer's algorithm. In other words, this reconstruction algorithm learns 3D shape with temporal smoothing, while learning the temporal smoothness term.

## 5   Experiments

We compared our algorithm with the iterative SFM algorithm presented by Torresani et al. [13], which we will refer to as ILSQ (iterative least-squares) in the following discussion[3]. ILSQ optimizes Equations 2 and 3 by alternating optimization of each of the unknowns (rotation, basis shapes, and coefficients). We also improved the algorithm by updating the translations as well. When some data is missing, ILSQ optimizes with respect to the available data. For both algorithms, the rigid motion is initialized by Tomasi-Kanade [12], and random initialization of the shape basis and coefficients. For the algorithm presented in section 3, we adopted an annealing scheme that forces $\sigma^2$ to remain large in the initial steps of the optimization. We refer to our new algorithms as EM-Gaussian and EM-LDS.

We tested the algorithms on a synthetic animation of a deforming shark in Figure 1. The motion consists of rigid rotation plus deformations generated by $K = 2$ basis shapes. The average reconstruction errors in Z for ILSQ and EM-Gaussian are respectively 7.10% and 2.50% on this sequence after 100 parameter updates.[4] By enforcing temporal smoothness

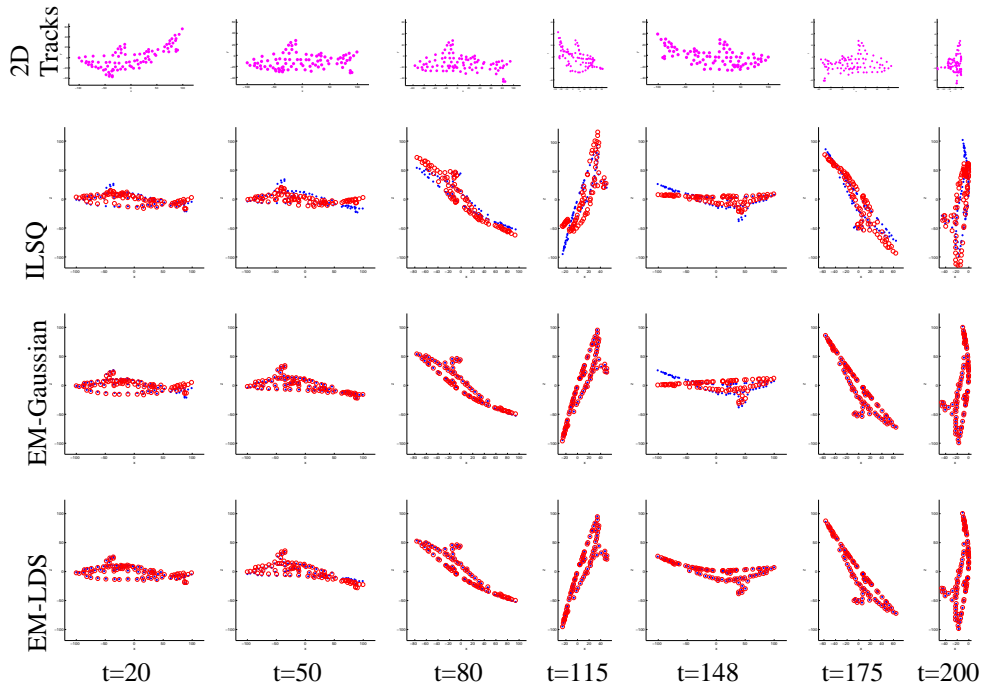

Figure 1: Reconstructions of the shark sequence using the three algorithms. Each algorithm was given 2D tracks as inputs; reconstructions are shown here from a different viewpoint than the inputs to the algorithm. Ground-truth features are shown as blue dots; reconstructions are red circles. Note that, although ILSQ gets approximately the correct shape in most cases, it misses details, whereas EM gives very accurate results most of the time. Some of the deformation errors of EM-Gaussian (e.g. for t=148) are corrected by EM-LDS through temporal smoothing.

EM-LDS was able to correct some of the deformation errors of EM-Gaussian. The average Z error for EM-LDS on the shark sequence after 100 EM iterations is 1.24%. Videos of the shark reconstructions and the Matlab software used for these experiments are available from `http://movement.stanford.edu/learning-nr-shape/`.

In highly-constrained cases — low-rank motion, no image noise, and no missing data — ILSQ achieved reasonably good results. However, EM-Gaussian gave better results in nearly every case, and dramatically better results in underconstrained cases. Figure 2(a) and (b) show experimental results on another set of artificial data consisting of random basis shapes. Figure 2(a) shows the results of reconstruction with missing data; the ILSQ results degrade much faster as the percentage of missing data increases. Figure 2(b) shows the effect of changing the complexity of the model, while leaving the complexity of the data fixed. ILSQ yields poor results when the model complexity does not closely match the data complexity, but EM-Gaussian yields reasonable results regardless.

## 6  Discussion and future work

We have described an approach to non-rigid structure-from-motion with a probabilistic deformation model, and demonstrated its usefulness in the case of a Gaussian deformation model. We expect that more sophisticated distributions can be used to model more complex non-rigid shapes in video. More general graphical models with other correlations (such as from audio data) could be built from this method. Our method is also applicable to

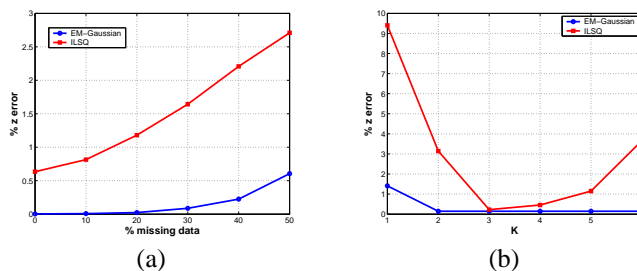

<table>
<tr><td>(a)</td><td>(b)</td></tr>
</table>

Figure 2: Error comparison between ILSQ and EM-Gaussian on random basis shapes. (a) Increasing missing data. As the percentage of missing feature tracks per frame increases, ILSQ degenerates much more rapidly than EM-Gaussian. (b) ILSQ gives poor results when the model complexity does not match the actual data complexity, whereas EM-Gaussian is relatively robust to this.

separating rigid from non-rigid motion in fully-observed data, as in Soatto and Yezzi's work [11]. Our models could easily be generalized to perspective projection, although the optimization may be more difficult.

**Acknowledgements.** Thanks to Hrishikesh Deshpande for assisting with an early version of this project, and to Stefano Soatto for discussing deformation ambiguities. Portions of this work were performed while LT was visiting New York University, AH was at University of Washington, and while CB was at Stanford University. LT and CB were supported by ONR grant N00014-01-1-0890 under the MURI program. AH was supported in part by UW Animation Research Labs, NSF grant IIS-0113007, the Connaught Fund, and an NSERC Discovery Grant.

## Footnotes

[1]In their work, Soatto and Yezzi address a slightly simpler problem where the 3D data is observed without noise or projection, and thus there are no weights to specify in this case

[2] The vec operator stacks the columns of a matrix into a vector, e.g. $\mathrm{vec}\left( \begin{bmatrix} a_0 & a_2 \\ a_1 & a_3 \end{bmatrix} \right) = [a_0, a_1, a_2, a_3]^T$. The operator is linear: $\mathrm{vec}(\mathbf{A} + \mathbf{B}) = \mathrm{vec}(\mathbf{A}) + \mathrm{vec}(\mathbf{B}), \mathrm{vec}(\alpha \mathbf{A}) = \alpha \mathrm{vec}(\mathbf{A})$ for any matrices $\mathbf{A}$ and $\mathbf{B}$ and scalar $\alpha$.

[3]In our experience, ILSQ always performs better than the algorithm of Bregler et al. [4].

[4]All errors are computed in percentage points: the average distance of the reconstructed point to the correct point divided by the size of the shape.

## References

[1] A. Blake and M. Isard. *Active Contours*. Springer-Verlag, 1998.

[2] V. Blanz and T. Vetter. A Morphable Model for the Synthesis of 3D Faces. In *Proceedings of SIGGRAPH 99*, Computer Graphics Proceedings, pages 187–194, Aug. 1999.

[3] M. Brand. Morphable 3D models from video. In *Proc. CVPR 2001*, 2001.

[4] C. Bregler, A. Hertzmann, and H. Biermann. Recovering Non-Rigid 3D Shape from Image Streams. In *Proc. CVPR 2000*, 2000.

[5] T. F. Cootes and C. J. Taylor. Statistical models of appearance for medical image analysis and computer vision. In *Proc. SPIE Medical Imaging*, 2001.

[6] Z. Ghahramani and G. E. Hinton. The EM Algorithm for Mixtures of Factor Analyzers. Technical Report CRG-TR-96-1, University of Toronto, 1996.

[7] A. Gruber and Y. Weiss. Factorization with Uncertainty and Missing Data: Exploiting Temporal Coherence. In *Proc. NIPS 2003*, 2003. In these proceedings.

[8] D. W. Jacobs. Linear Fitting with Missing Data for Structure-From-Motion. *Computer Vision and Image Understanding*, 82:57–82, 2001.

[9] H. Shum, K. Ikeuchi, and R. Reddy. Principal Component Analysis with Missing Data and Its Applications to Polyhedral Object Modeling. *IEEE Trans. PAMI*, 17(9):854–867, 1995.

[10] R. H. Shumway and D. S. Stoffer. An approach to time series smoothing and forecasting using the em algorithm. *J. Time Series Analysis*, 3(4):253–264, 1982.

[11] S. Soatto and A. J. Yezzi. Deformotion: Deforming Motion, Shape Averages, and the Joint Registration and Segmentation of Images. In *Proc. ECCV 2002*, May 2002.

[12] C. Tomasi and T. Kanade. Shape and motion from image streams under orthography: A factorization method. *Int. J. of Computer Vision*, 9(2):137–154, 1992.

[13] L. Torresani, D. Yang, G. Alexander, and C. Bregler. Tracking and Modeling Non-Rigid Objects with Rank Constraints. In *Proc. CVPR*, 2001.